# Object Classification from a Single Example Utilizing Class Relevance Metrics

**Michael Fink**
Interdisciplinary Center for Neural Computation
The Hebrew University, Jerusalem 91904, Israel
fink@huji.ac.il

## Abstract

We describe a framework for learning an object classifier from a single example. This goal is achieved by emphasizing the relevant dimensions for classification using available examples of related classes. Learning to accurately classify objects from a single training example is often unfeasible due to overfitting effects. However, if the instance representation provides that the distance between each two instances of the same class is smaller than the distance between any two instances from different classes, then a nearest neighbor classifier could achieve perfect performance with a single training example. We therefore suggest a two stage strategy. First, learn a metric over the instances that achieves the distance criterion mentioned above, from available examples of *other* related classes. Then, using the single examples, define a nearest neighbor classifier where distance is evaluated by the learned class relevance metric. Finding a metric that emphasizes the relevant dimensions for classification might not be possible when restricted to linear projections. We therefore make use of a kernel based metric learning algorithm. Our setting encodes object instances as sets of locality based descriptors and adopts an appropriate image kernel for the class relevance metric learning. The proposed framework for learning from a single example is demonstrated in a synthetic setting and on a character classification task.

## 1 Introduction

We describe a framework for learning to accurately discriminate between two target classes of objects (e.g. platypuses and opossums) using a single image of each class. In general, learning to accurately classify object images from a single training example is unfeasible due to overfitting effects of high dimensional data. However, if a certain distance function over the instances guarantees that all within-class distances are smaller than any between-class distance, then nearest neighbor classification could achieve perfect performance with a single training example. We therefore suggest a two stage method. First, learn from available examples of other related classes (like beavers, skunks and marmots), a metric over the instance space that satisfies the distance criterion mentioned above. Then, define a nearest neighbor classifier based on the single examples. This nearest neighbor classifier calculates distance using the class relevance metric.

The difficulty in achieving robust object classification emerges from the instance variety of object appearance. This variability results from both class relevant and class non-relevant dimensions. For example, adding a stroke crossing the digit 7, adds variability due to a class relevant dimension (better discriminating 7's from 1's), while italic writing adds variability in a class irrelevant dimension. Often certain non-relevant dimensions could be avoided by the designer's method of representation (e.g. incorporating translation invariance). Since such guiding heuristics may be absent or misleading, object classification systems often use numerous positive examples for training, in an attempt to manage within class variability. We are guided by the observation that in many settings providing an extended training set of certain classes might be costly or impossible due to scarcity of examples, thus motivating methods that suffice with few training examples.

Categories' appearance variety seems to inherently entail severe overfitting effects when only a small sample is available for training. In the extreme case of learning from a single example it appears that the effects of overfitting might prevent any robust category generalization. These overfitting effects tend to exacerbate as a function of the representation dimensionality.

In the spirit of the learning to learn literature [17], we try to overcome the difficulties that entail training from a single example by using available examples from several other related objects. Recently, it has been demonstrated that objects share distribution densities on deformation transforms [13], shape or appearance [6]; and that objects could be detected by a common set of reusable features [1, 18]. We suggest that in many visual tasks it is natural to assume that one common set of constraints characterized a common set of relevant and non-relevant dimensions shared by a specific family of related classes [10].

Our paper is organized as follows. In Sec. 2 we start by formalizing the task of training from a single example. Sec. 3 describes a kernel over sets of local features. We then describe in Sec. 4 a kernel based method for learning a pseudo-metric that is capable of emphasizing the relevant dimensions and diminishing the overfitting effects of non-relevant dimensions. By projecting the single examples using this class relevance pseudo-metric, learning from a single example becomes feasible. Our experimental implementation described in Sec. 5, adopts shape context descriptors [3] of Latin letters to demonstrate the feasibility of learning from a single example. We conclude with a discussion on the scope and limitations of the proposed method.

## 2   Problem Setting

Let $\mathcal{X}$ be our object instance space and let $u$ and $v$ indicate two classes defined over $\mathcal{X}$. Our goal is to generate a classifier $h(\mathbf{x})$ which discriminates between instances of the two object classes $u$ and $v$. Formally, $h : \mathcal{X} \rightarrow \{u, v\}$ so that $\forall \mathbf{x}$ in class $u$, $h(\mathbf{x}) = u$ and $\forall \mathbf{x}$ in class $v$, $h(\mathbf{x}) = v$. We adopt a local features representation for encoding object images. Thus, every $\mathbf{x}$ in our instance space is characterized by the set $\{l_j^i, p_j^i\}_{j=1}^k$ where $l_j^i$ is a locality based descriptor calculated at location $p_j^i$ of image $i$ [1]. We assume that $l_j^i$ is encoded as a vector of length $n$ and that the same number of locations $k$ are selected from each image[2]. Thus any $\mathbf{x}$ in our instance space $\mathcal{X}$ is defined by an $n \times k$ matrix.

Our method uses a single instance from classes $u$ and $v$ as well as instances from other related classes. We denote by $q$ the total number of classes. An example is formally defined as a pair $(\mathbf{x}, y)$ where $\mathbf{x} \in \mathcal{X}$ is an instance and $y \in \{1, \ldots, q\}$ is the index of the instance's class. The proposed setting postulates that two sets are provided for training $h(\mathbf{x})$:

- A single example of class $u$, $(\mathbf{x}, u)$ and a single example of class $v$, $(\mathbf{x}, v)$
- An extended sample $\{(\mathbf{x}_1, y_1), \ldots, (\mathbf{x}_m, y_m)\}$ of $m \gg 1$ examples where $\mathbf{x}_i \in \mathcal{X}$ and $y_i \notin \{u, v\}$ for all $1 \leq i \leq m$.

We say that a set of classes is $\gamma > 0$ *separated* with respect to a distance function $d$ if for any pair of examples belonging to the same class $\{(\mathbf{x}_1, c), (\mathbf{x}_1', c)\}$, the distance $d(\mathbf{x}_1, \mathbf{x}_1')$ is smaller than the distance between any pair of examples from different classes $\{(\mathbf{x}_2, e), (\mathbf{x}_2', g)\}$ by at least $\gamma$:

$$d(\mathbf{x}_1, \mathbf{x}_1') \leq d(\mathbf{x}_2, \mathbf{x}_2') - \gamma .$$

Recall that our goal is to generate a classifier $h(\mathbf{x})$ which discriminates between instances of the two object classes $u$ and $v$. In general, learning from a single example is prone to overfitting, yet if a set of classes is $\gamma$ separated, a single example is sufficient for a nearest neighbor classifier to achieve perfect performance. Therefore our proposed framework is composed of two stages:

1. Learn from the extended sample a distance function $d$ that achieves $\gamma$ separation on classes $y \notin \{u, v\}$.
2. Learn a nearest neighbor classifier $h(\mathbf{x})$ from the single examples, where the classifier employs $d$ for evaluating distances.

From the theory of large margin classifiers we know that if a classifier achieves a large margin separation on an i.i.d. sample then it is likely to generalize well. We informally state that analogously, if we find a distance function $d$ such that $q - 2$ classes that form the extended sample are separated by a large $\gamma$ with respect to $d$, with high probability classes $u$ and $v$ should exhibit the separation characteristic as well. If these assumptions hold and $d$ indeed induces $\gamma$ separation on classes $u$ and $v$, then a nearest neighbor classifier would generalize well from a single training example of the target classes. It should be noted that when training from a single example nearest neighbor, max margin and naive Bayes algorithms, all yield the same classification rule. For simplicity we choose to focus on a nearest neighbor formulation. We will later show how the distance $d$ might be parameterized by measuring Euclidian distance, after applying a linear projection $W$ to the original instance space. Classifying instances in the original instance space by comparing them to the target classes' single examples $\mathbf{x}$ and $\mathbf{x}'$, leads to overfitting. In contrast, our approach projects the instance space by $W$ and only then applies a nearest neighbor distance measurement to the projected single examples $W\mathbf{x}$ and $W\mathbf{x}'$. Our method relies on the distance $d$, parameterized by $W$, to achieve $\gamma$ separation on classes $u$ and $v$. In certain problems it is not possible to achieve $\gamma$ separation by using a distance function which is based on a linear transformation of the instance space. We therefore propose to initially map the instance space $\mathcal{X}$ into an implicit feature space defined by a Mercer kernel [20].

## 3   A Principal Angles Image Kernel

We dedicate this section to describe a Mercer kernel between sets of locality based image features $\{l_j^i, p_j^i\}_{j=1}^k$ encoded as $n \times k$ matrices. Although potentially advantageous in many applications, one shortcoming in adopting locality based feature descriptors lays in the vagueness of matching two sets of corresponding locations $p_j^i$, $p_{j'}^{i'}$ selected from different object images $i$ and $i'$ (see Fig. 1). Recently attempts have been made to tackle this problem [19], we choose to follow [20] by adopting the principal angles kernel approach that implicitly maps $\mathbf{x}$ of size $n \times k$ to a significantly higher $\binom{n}{k}$-dimensional feature space $\phi(\mathbf{x}) \in F$. The principal angles kernel is formally defined as:

$$K(\mathbf{x}_i, \mathbf{x}_{i'}) = \phi(\mathbf{x}_i)\phi(\mathbf{x}_{i'}) = det(Q_i^\top Q_{i'})^2$$

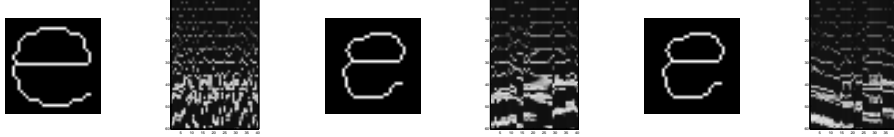

Figure 1: The 40 columns in each matrix encode 60-dimentional descriptors (detailed in Sec. 5) of three instances of the letter $e$. Although the objects are similar, the random sequence of sampling locations $p_j^i$ entails column permutation, leading to apparently different matrices. Ignoring selection permutation by reshaping the matrices as vectors would further obscure the relevant similarity. A kernel applied to matrices that is invariant to column permutation can circumvent this problem.

The columns of $Q_i$ and $Q_{i'}$ are each an orthonormal basis resulting from a QR decomposition of the instances $\mathbf{x}_i$ and $\mathbf{x}_{i'}$ respectively. One advantage of the principal angels kernel emerges from its invariance to column permutations of the instance matrices $\mathbf{x}_i$ and $\mathbf{x}_{i'}$, thus circumventing the location matching problem stated above. Extensions of the principal angles kernel that have the additional capacity to incorporate knowledge on the accurate location matching, might enhance the kernel's descriptive power [16].

## 4   Learning a Class Relevance Pseudo-Metric

In this section we describe the two stage framework for learning from a single example to accurately classify classes $u$ and $v$. We focus on transferring information from the extended sample of classes $y \notin \{u, v\}$ in the form of a learned pseudo-metric over $\mathcal{X}$. For sake of clarity we will start by temporarily referring to the instance space $\mathcal{X}$ as a vector space, but later return to our original definition of instances in $\mathcal{X}$ as being matrices which columns encode a selected set of locality based descriptors $\{l_j^i, p_j^i\}_{j=1}^k$.

A pseudo-metric is a function $d : \mathcal{X} \times \mathcal{X} \to \mathbb{R}$, which satisfies three requirements, (i) $d(\mathbf{x}, \mathbf{x}') \geq 0$, (ii) $d(\mathbf{x}, \mathbf{x}') = d(\mathbf{x}', \mathbf{x})$, and (iii) $d(\mathbf{x}_1, \mathbf{x}_2) + d(\mathbf{x}_2, \mathbf{x}_3) \geq d(\mathbf{x}_1, \mathbf{x}_3)$. Following [14], we restrict ourselves to learning pseudo-metrics of the form

$$d_A(\mathbf{x}, \mathbf{x}') \equiv \sqrt{(\mathbf{x} - \mathbf{x}')^\top A (\mathbf{x} - \mathbf{x}')} \ ,$$

where $A \succeq 0$ is a symmetric positive semi-definite (PSD) matrix.

Since $A$ is PSD, there exists a matrix $W$ such that

$$(\mathbf{x} - \mathbf{x}')^\top A (\mathbf{x} - \mathbf{x}') = \|W\mathbf{x} - W\mathbf{x}'\|_2^2 \ .$$

Therefore, $d_A(\mathbf{x}, \mathbf{x}')$ is the Euclidean distance between the image of $\mathbf{x}$ and $\mathbf{x}'$ due to a linear transformation $W$. We now restate our goal as that of using the extended sample of classes $y \notin \{u, v\}$ in order to find a linear projection $W$ that achieves $\gamma$ separation by emphasizing the relevant dimensions for classification and diminishing the overfitting effects of non-relevant dimensions.

Several linear methods exist for finding a class relevance projection [2, 9], some of which have a kernel based variant [12]. Our method of choice, proposed by [14], is an online algorithm characterized by its capacity to efficiently handle high dimensional input spaces. In addition the method's margin based approach is directly aimed at achieving our $\gamma$ separation goal. We convert the online algorithm for finding $A$ to our batch setting by averaging the resulting $A$ over the algorithm's $\tau$ iterations [4].

Fig. 2 demonstrates how a class relevance pseudo-metric enables training a nearest neighbor classifier from a single example of two classes in a synthetic two dimensional setting.

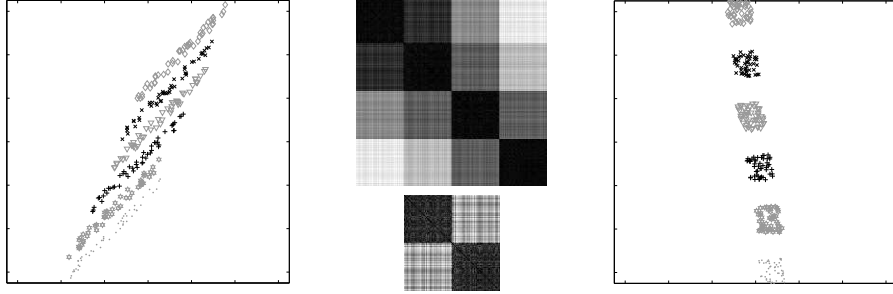

Figure 2: A synthetic sample of six obliquely oriented classes in a two dimensional space (left). A class relevance metric is calculated from the ($m = 200$) examples of the four classes $y \notin \{u, v\}$ marked in gray. The examples of the target classes $u$ and $v$, indicated in black, are *not* used in calculating the metric. After learning the pseudo-metric, all the instances of the six classes are projected to the class relevance space. Here distance measurements are performed between the four classes $y \notin \{u, v\}$. The results are displayed as a color coded distance matrix (center-top). Throughout the paper distance matrix indices are ordered by class so $\gamma$ separated classes should appear as block diagonal matrices. Although not participating in calculating the pseudo-metric, classes $u$ and $v$ exhibit $\gamma$ separation (center-bottom). After the class relevance projection, a nearest neighbor classifier will generalize well from any single example of classes $u$ and $v$ (right).

In the primal setting of the pseudo-metric learning, we temporarily addressed our instances $\mathbf{x}$ as vectors, thus enabling subtraction and dot product operations. These operations have no clear interpretation when applied to our representation of objects as sets of locality based descriptors $\{l_j^i, p_j^i\}_{j=1}^k$. However the adopted pseudo-metric learning algorithm has a dual version, where interface to the data is limited to inner products. In the dual mode $A$ is implicitly represented by a set of support examples $\{\mathbf{x}_j\}_{j=1}^\tau$ and by learning two sets of scalar coefficients $\{\beta_h\}_{h=1}^f$ and $\{\rho_{j,h}\}_{(j,h)=(1,1)}^{(\tau,f)}$. Thus, applying the dual representation of the pseudo-metric, distances between instances $\mathbf{x}$ and $\mathbf{x}'$ could be calculated by:

$$d_A(\mathbf{x}, \mathbf{x}')^2 \quad = \quad \sum_{h=1}^f \beta_h \left( \sum_{j=1}^\tau \rho_{j,h} \left[ K(\mathbf{x}_j, \mathbf{x}) - K(\mathbf{x}_j, \mathbf{x}') - K(\mathbf{x}'_j, \mathbf{x}) + K(\mathbf{x}'_j, \mathbf{x}') \right] \right)^2$$

$d_A(\mathbf{x}, \mathbf{x}')^2$ in the above equation is therefore evaluated by calling upon the principal angles kernel previously described in Sec. 3. Fig. 3 demonstrates how a class relevance pseudo-metric enables training from a single example in a classification problem, where nonlinear projection of the instance space is required for achieving a $\gamma$ margin.

## 5  Experiments

Sets of six lowercase Latin letters (i.e. *e, n, t, f, h* and *c*) are selected as target classes for our experiment (see examples in Fig. 4). The Latin character database [7] includes 60 examples of each letter. Two representations are examined. The first is a pixel based representation resulting from column-wise encoding the raw $36 \times 36$ pixel images as a vector of length 1296. Our second representation adopts the shape context descriptors for object encoding. This representation relies on a set of 40 locations $p_j$ randomly sampled from the object contour. The descriptor of each location $p_j$ is based on a 60-bin histogram (5 radius $\times$ 12 orientation bins) summing the number of "lit" pixels falling in each specific radius and orientation bin (using $p_j$ as the origin). Each example in our instance space is therefore encoded as a $60 \times 40$ matrix. Three shape context descriptors are depicted in Fig. 4. Shape

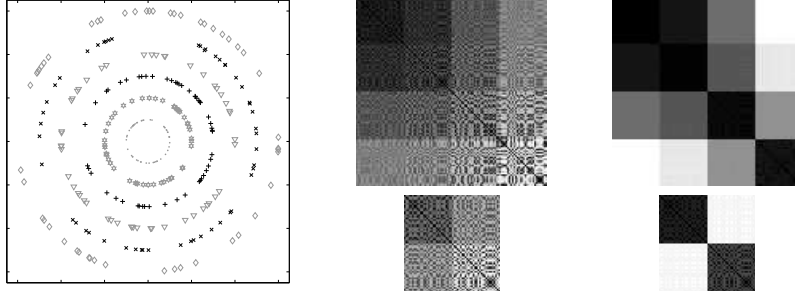

Figure 3: A synthetic sample of six co-centric classes in a two dimensional space (left). Two class relevance metrics are calculated from the examples ($m = 200$) of the four classes $y \notin \{u, v\}$ marked in gray using either a linear or a second degree polynomial kernel. The examples of the target classes $u$ and $v$, indicated in black, are *not* used in calculating the metrics. After learning both metrics, all the instances of the six classes are projected using both class relevance metrics. Then distance measurements are performed between the four classes $y \notin \{u, v\}$. The resulting linear distance matrix (center-top) and polynomial distance matrix (right-top) seem qualitatively different. Classes $u$ and $v$, not participating in calculating the pseudo-metric, exhibit $\gamma$ separation only when using an appropriate kernel (right-bottom). A linear kernel cannot accommodate $\gamma$ separation between co-centric classes (center-bottom).

context descriptors have proven to be robust in many classification tasks [3] and avoid the common reliance on a detection of (the often elusive) interest points. In many writing systems letters tend to share a common underlying set of class relevant and non-relevant dimensions (Fig. 5-left). We therefore expect that letters should be a good candidate for exhibiting that a class relevance pseudo-metric achieving a large margin $\gamma$, would induce the distance separation characteristic on two additional letter classes in the same system.

We randomly select a single example of two letters (i.e. *e* and *n*) for training and save the remaining examples for testing. A nearest neighbor classifier is defined by the two examples, in order to assess baseline performance of training from a single example. A linear kernel is applied for the pixel based representation while the principal angles kernel is used for the shape context representation. Performance is assessed by averaging the generalization accuracy (on the unseen testing examples) over 900 repetitions of random letter selection. Baseline results for the shape context and pixel representations are depicted in Fig. 5 A and C, respectively (letter references to Fig. 5 appear on the right bar plot).

We now make use of the 60 examples of each of the remaining letters (i.e. *t, f, h* and *c*) in order to learn a distance over letters. The dual formulation of the pseudo-metric learning algorithm (described in Sec. 4) is implemented and run for 1000 iterations over random pairs selected from the 240 training examples ($m = 4$ classes $\times$ 60 examples). The same 900 example pairs used in the baseline testing are now projected using the letter metric. It is observed that the learned pseudo-metric approximates the separation goal on the two unseen target classes $u$ and $v$ (center plot of Fig. 5). A nearest neighbor classifier is then trained using the projected examples ($W\mathbf{x}, W\mathbf{x}'$) from class $u$ and $v$. Performance is assessed as in the baseline test. Results for the shape context based representation are presented in Fig. 5B while performance of the pixel based representation is depicted in Fig. 5E.

When training from a single example the lower dimensional pixel representation (of size 1296) displays less of an overfitting effect than the shape context representation paired with a principal angles kernel (implicitly mapped by the kernel from size $60 \times 40$ to size $\binom{60}{40}$). This effect could be seen when comparing Fig. 5D and Fig. 5A. It is not surprising that although some dimensions in the high dimensional shape context feature represen-

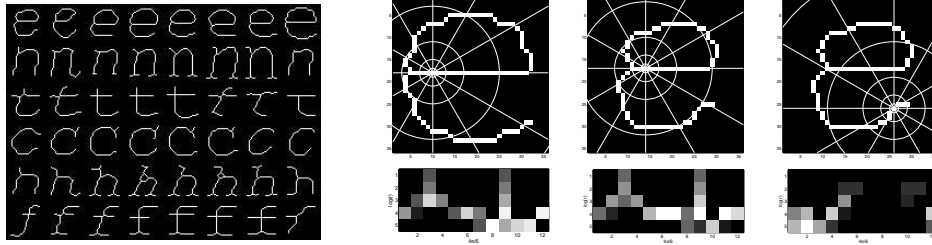

Figure 4: Examples of six character classes used in the letter classification experiment (left). The context descriptor at location $p$ is based on a 60-bin histogram (5 radius $\times$ 12 orientation bins) of all surrounding pixels, using $p$ as the origin. Three examples of the letter e, depicted with the histogram bin boundaries (top) and three derived shape context histograms plotted as $log(radius) \times orientation$ bins (bottom). Note the similarity of the two shape context descriptors sampled from analogous locations on two different examples of the letter $e$ (two bottom-center plots). The shape context of a descriptor sampled from a distant location is evidently different (right).

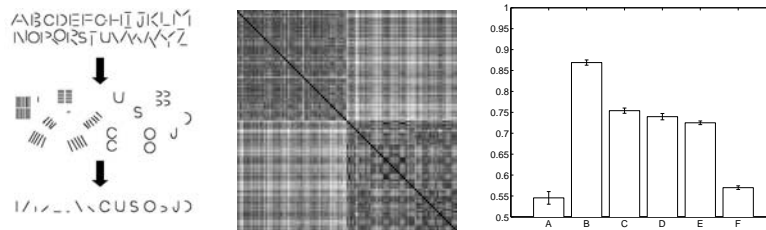

Figure 5: Letters in many writing systems, like uppercase Latin, tend to share a common underlying set of class relevant and non-relevant dimensions (left plot adapted from [5]). A class relevance pseudo-metric was calculated from four letters (i.e. *t, f, h* and *c*). The central plot depicts the distance matrix of the two target letters (i.e. *e* and *n*) after the class relevance pseudo-metric projection. The right plot presents average accuracy of classifiers trained on a single example of lowercase letters (i.e. *e* and *n*) in the following conditions: A. Shape Context Representation B. Shape Context Representation after class relevance projection C. Shape Context Representation after a projection derived from uppercase letters D. Pixel Representation E. Pixel Representation after class relevance projection F. Pixel Representation after a projection derived from uppercase letters.

tation might exhibit superior performance in classification, increasing the representation dimensionality introduces numerous non-relevant dimensions, thus causing the substantial overfitting effects displayed at Fig. 5A. However, it appears that by projecting the single examples using the class relevance pseudo-metric, the class relevant dimensions are emphasized and hindering effects of the non-relevant dimensions are diminished (displayed at Fig. 5B). It should be noted that a simple linear pseudo-metric projection cannot achieve the desired margin on the extended sample, and therefore seems not to generalize well from the single trial training stage. This phenomenon is manifested by the decrease in performance when linearly projecting the pixel based representation (Fig. 5E).

Our second experiment is aimed at examining the underlying assumptions of the proposed method. Following the same setting as in the first experiment we randomly selected two lowercase Latin letters for the single trial training task, while applying a pseudo-metric projection derived from uppercase Latin letters. It is observed that utilizing a less relevant pseudo-metric attenuates the benefit in the setting based on the shape context representation paired with the principal angles kernel (Fig. 5C). In the linear pixel based setting projecting lowercase letters to the uppercase relevance directions significantly deteriorates performance (Fig. 5F), possibly due to deemphasizing the lowercase characterizing curves.

# 6  Discussion

We proposed a two stage method for classifying object images using a single example. Our approach, first attempts to learn from available examples of other related classes, a class relevance metric where all within class distances are smaller than between class distances. We then, define a nearest neighbor classifier for the two target classes, using the class relevance metric. Our high dimensional representation applied a principal angles kernel [20] to sets of local shape descriptors [3]. We demonstrated that the increased representational dimension aggravates overfitting when learning from a single example. However, by learning the class relevance metric from available examples of related objects, relevant dimensions for classification are emphasized and the overfitting effects of irrelevant dimensions are diminished. Our technique thereby generates a highly accurate classifier from only a single example of the target classes. Varying the choice of local feature descriptors [11, 15], and enhancing the image kernel [16] might further improve the proposed method's generalization capacity in other object classification settings. We assume that our examples represent a set of classes that originate from a common set of constraints, thus imposing that the classes tend to agree on the relevance and non-relevance of different dimensions. Our assumption holds well for objects like textual characters [5]. It has been recently demonstrated that feature selection mechanisms can enable real-world object detection by a common set of shared features [18, 8]. These mechanisms are closely related to our framework when considering the common features as a subset of directions in our class relevance pseudo-metric. We therefore aim our current research at learning to classify more challenging objects.

## Footnotes

[1] $p_j^i$ might be selected from image $i$ either randomly, or by a specialized interest point detector.

[2] This assumption could be relaxed as demonstrated in [16, 19]

# References

[1] S. Krempp, D. Geman and Y. Amit. Sequential learning of reusable parts for object detection. *Technical report, CS Johns Hopkins*, 2002.

[2] A. Bar-Hillel, T. Hertz, N. Shental and D. Weinshall. Learning Distance Functions Using Equivalence Relations. *Proc ICML03*, 2003.

[3] S. Belongie, J. Malik and J. Puzicha. Matching Shapes. *Proc. ICCV*, 2001.

[4] N. Cesa-Bianchi, A. Conconi, and C. Gentile. On the generalization ability of on-line learning algorithms. *IEEE Transactions on Information Theory. To appear* , 2004.

[5] M.A. Chanagizi and S. Shimojo. Complexity and redundancy of writing systems, and implications for letter perception. *under review*, 2004.

[6] L. Fei-Fei, R. Fergus and P. Perona. Learning generative visual models from few training examples. *CVPR04 Workshop on Generative-Model Based Vision*, 2004.

[7] M. Fink. A Latin Character Database. *www.cs.huji.ac.il/~fink*, 2004.

[8] M. Fink and K. Levi. Encoding Reusable Perceptual Features Enables Learning Future Categories from Few Examples. *Tech Report CS HUJI* , 2004.

[9] K. Fukunaga. Statistical Pattern Recognition. *San Diego: Academic Press 2nd Ed.*, 1990.

[10] K. Levi and M. Fink. Learning From a Small Number of Training Examples by Exploiting Object Categories. *LCVPR04 workshop on Learning in Computer Vision*, 2004.

[11] D. G. Lowe. Object recognition from local scale-invariant features. *Proc. ICCV99*, 1999.

[12] S. Mika, G. Ratsch, J. Weston, B. Scholkopf and K. R. Muller. Fisher Discriminant Analysis with Kernels. *Neural Networks for Signal Processing IX*, 1999.

[13] E. Miller, N. Matsakis and P. Viola. Learning from One Example through Shared Densities on Transforms. *Proc. CVPR00(1)*, 2000.

[14] S. Shalev, Y. Singer and A. Ng. Online and Batch Learning of Pseudo-Metrics. *Proc. ICML04*, 2004.

[15] M. J. Swain and D. H. Ballard. Color Indexing. *IJCV 7(1)*, 1991.

[16] A. Shashua and T. Hazan. Threading Kernel Functions: Localized vs. Holistic Representations and the Family of Kernels over Sets of Vectors with Varying Cardinality. *NIPS04 under review*.

[17] S. Thrun and L. Pratt. Learning to Learn. *Kluwer Academic Publishers*, 1997.

[18] A. Torralba, K. Murphy and W. Freeman. Sharing features: effi cient boosting procedures for multiclass object detection. *Proc. CVPR04*, 2004.

[19] C.Wallraven, B.Caputo and A.Graf Recognition with Local features kernel recipe. *ICCV*, 2003.

[20] L. Wolf and A. Shashua. Learning over sets using kernel principal angles. *JML 4*, 2003.
